# Network activity determines spatio-temporal integration in single cells

**Öjvind Bernander, Christof Koch** *
Computation and Neural Systems Program,
California Institute of Technology,
Pasadena, Ca 91125, USA.

**Rodney J. Douglas**
Anatomical Neuropharmacology Unit,
Dept. Pharmacology,
Oxford, UK.

## Abstract

Single nerve cells with static properties have traditionally been viewed as the building blocks for networks that show emergent phenomena. In contrast to this approach, we study here how the overall network activity can control single cell parameters such as input resistance, as well as time and space constants, parameters that are crucial for excitability and spatio-temporal integration. Using detailed computer simulations of neocortical pyramidal cells, we show that the spontaneous background firing of the network provides a means for setting these parameters. The mechanism for this control is through the large conductance change of the membrane that is induced by both non-NMDA and NMDA excitatory and inhibitory synapses activated by the spontaneous background activity.

## 1   INTRODUCTION

Biological neurons display a complexity rarely heeded in abstract network models. Dendritic trees allow for local interactions, attenuation, and delays. Voltage- and

time-dependent conductances can give rise to adaptation, burst-firing, and other non-linear effects. The extent of temporal integration is determined by the time constant, and spatial integration by the "leakiness" of the membrane. It is unclear which cell properties are computationally significant and which are not relevant for information processing, even though they may be important for the proper functioning of the cell. However, it is crucial to understand the function of the component cells in order to make relevant abstractions when modeling biological systems. In this paper we study how the spontaneous background firing of the network as a whole can strongly influence some of the basic integration properties of single cells.

## 1.1   Controlling parameters via background synaptic activity

The **input resistance**, $R_{in}$, is defined as $\frac{dV}{dI}$, where $dV$ is the steady state voltage change in response to a small current step of amplitude $dI$. $R_{in}$ will vary throughout the cell, and is typically much larger in a long, narrow dendrite than in the soma. However, the somatic input resistance is more relevant to the spiking behavior of the neuron, since spikes are initiated at or close to the soma, and hence $R_{in,soma}$ (henceforth simply referred to as $R_{in}$) will tell us something of the sensitivity of the cell to charge reaching the soma.

The **time constant**, $\tau_m$, for a passive membrane patch is $R_m \cdot C_m$, the membrane resistance times the membrane capacitance. For membranes containing voltage-dependent non-linearities, exponentials are fitted to the step response and the largest time constant is taken to be the membrane time constant. A large time constant implies that any injected charge leaks away very slowly, and hence the cell has a longer "memory" of previous events.

The parameters discussed above ($R_{in}$, $\tau_m$) clearly have computational significance and it would be convenient to be able to change them dynamically. Both depend directly on the membrane conductance $G_m = \frac{1}{R_m}$, so any change in $G_m$ will change the parameters. Traditionally, however, $G_m$ has been viewed as static, so these parameters have also been considered static. How can we change $G_m$ dynamically?

In traditional models, $G_m$ has two components: active (time- and voltage-dependent) conductances and a passive "leak" conductance. Synapses are modeled as conductance changes, but if only a few are activated, the cable structure of the cell will hardly change at all. However, it is well known that neocortical neurons spike spontaneously, in the absence of sensory stimuli, at rates from 0 to 10 $Hz$. Since neocortical neurons receive on the order of $5,000$ to $15,000$ excitatory synapses (Larkman, 1991), this spontaneous firing is likely to add up to a large total conductance (Holmes & Woody, 1989). This synaptic conductance becomes crucial if the non-synaptic conductance components are small. Recent evidence show indeed that the non-synaptic conductances are relatively small (when the cell is not spiking) (Anderson et al., 1990). Our model uses a leak $R_m = 100,000\ k\Omega cm^2$, instead of more conventional values in the range of $2,500$–$10,000\ k\Omega cm^2$. These two facts, high $R_m$ and synaptic background activity, allow $R_{in}$ and $\tau_m$ to change by more than ten-fold, as described below in this paper.

## 2  MODEL

A typical layer V pyramidal cell (fig. 2) in striate cortex was filled with HRP during *in vivo* experiments in the anesthetized, adult cat (Douglas et al., 1991). The 3-D coordinates and diameters of the dendritic tree were measured by a computer-assisted method and each branch was replaced by a single equivalent cylinder. This morphological data was fed into a modified version of NEURON, an efficient single cell simulator developed by Hines (1989). The dendrites were passive, while the soma contained seven active conductances, underlying spike generation, adaptation, and slow onset for weak stimuli. The model included two sodium conductances (a fast spiking current and a slower non-inactivating current), one calcium conductance, and four potassium conductances (delayed rectifier, slow 'M' and 'A' type currents, and a calcium-dependent current). The active conductances were modeled using a Hodgkin-Huxley-like formalism.

The model used a total of 5,000 synapses. The synaptic conductance change in time was modeled with an alpha function, $g(t) = \frac{g_{peak}e}{t_{peak}}te^{-t/t_{peak}}$. 4,000 synapses were fast excitatory non-NMDA or AMPA-type ($t_{peak} = 1.5\ msec$, $g_{peak} = 0.5\ nS$, $E_{rev} = 0\ mV$), 500 were medium-slow inhibitory $GABA_A$ ($t_{peak} = 10\ msec$, $g_{peak} = 1.0\ nS$, $E_{rev} = -70\ mV$), and 500 were slow inhibitory $GABA_B$ ($t_{peak} = 40\ msec$, $g_{peak} = 0.1\ nS$, $E_{rev} = -95\ mV$). The excitatory synapses were less concentrated towards the soma, while the inhibitory ones were more so. For a more detailed description of the model, see Bernander et al. (1991).

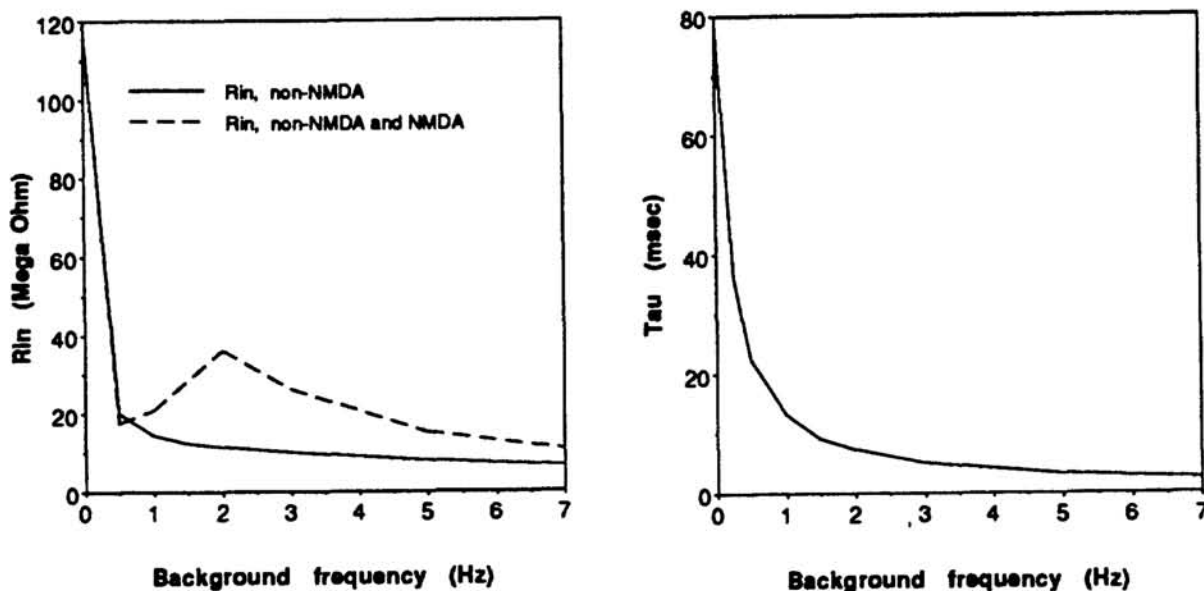

Figure 1: **Input resistance and time constant as a function of background frequency.** In (a), the solid line corresponds to the "standard" model with passive dendrites, while the dashed line includes active NMDA synapses as described in the text.

## 3   RESULTS

### 3.1   $R_{in}$ and $\tau_m$ change with background frequency

Fig. 1 illustrates what happens to $R_{in}$ and $\tau_m$ when the synaptic background activities of all synaptic types are varied simultaneously. In the absence of any synaptic input, $R_{in} = 110 \ M\Omega$ and $\tau_m = 80 \ msec$. At 1 $Hz$ background activity, on average 5 synaptic events are impinging on the cell every $msec$, contributing a total of 24 $nS$ to the somatic input conductance $G_{in}$. Because of the reversal potential of the excitatory synapses (0 $mV$), the membrane potential throughout the cell is pulled towards more depolarizing potentials, activating additional active currents. Although these trends continue as $f$ is increased, the largest change can be observed between 0 and 2 $Hz$.

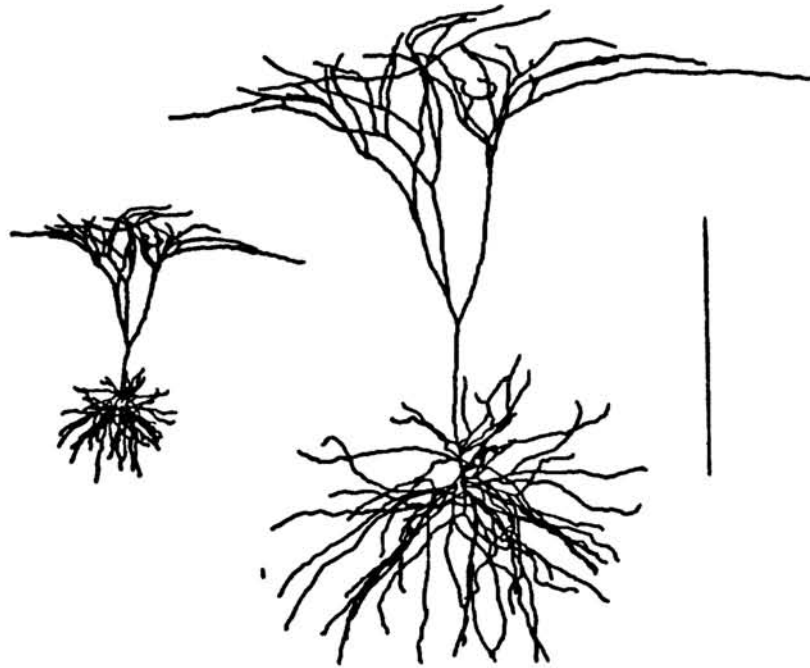

**Figure 2: Spatial integration as a function of background frequency.**
Each dendrite has been "stretched" so that its apparent length corresponds to its electrotonic length. The synaptic background frequency was 0 $Hz$ (left) and 2 $Hz$ (right). The scale bar corresponds to 1 $\lambda$ (length constant).

Activating synaptic input has two distinct effects: the conductance of the postsynaptic membrane increases and the membrane is depolarized. The system can, at least in principle, independently control these two effects by differentially varying the spontaneous firing frequencies of excitatory versus inhibitory inputs. Thus, increasing $f$ selectively for the $GABA_B$ inhibition will further increase the membrane conductance but move the resting potential towards more hyperpolarizing

potentials.

Note that the 0 $Hz$ case corresponds to experiments made with *in vitro* slice preparations or culture. In this case incoming fibers have been cut off and the spontaneous firing rate is very small. Careful studies have shown very large values for $R_{in}$ and $\tau_m$ under these circumstances (e.g. Spruston & Johnston, 1991). *In vivo* preparations, on the other hand, leave the cortical circuitry intact and much smaller values of $R_{in}$ and $\tau_m$ are usually recorded.

## 3.2   Spatial integration

Varying synaptic background activity can have a significant impact on the electrotonic structure of the cell (fig. 2). We plot the electrotonic distance of any particular point from the cell body, that is the sum of the electrotonic length's $L_i = \sum_j (l_j/\lambda_j)$ associated with each dendritic segment $i$, where $\lambda_j = \sqrt{\frac{R_m \cdot d_j}{4 \cdot R_i}}$ is the electrotonic length constant of compartment $j$, $l_j$ its anatomical length and the sum is taken over all compartments between the soma and compartment $i$.

Increasing the synaptic background activity from $f = 0$ to $f = 2\ Hz$ has the effect of stretching the "distance" of any particular synapse to the soma by a factor of about 3, on average. Thus, while a distal synapse has an associated $L$ value of about 2.6 at 2 $Hz$ it shrinks to 1.2 if all network activity is shut off, while for a synapse at the tip of a basal dendrite, $L$ shrinks from 0.7 to 0.2. In fact, the EPSP induced by a single excitatory synapse at that location goes from 39 to 151 $\mu V$, a decrease of about 4. Thus, when the overall network activity is low, synapses in the superficial layer of cortex could have a significant effect on somatic discharge, while having only a weak modulatory effect on the soma if the overall network activity is high. Note that basal dendrites, which receive a larger number of synapses, stretch more than apical dendrites.

## 3.3   Temporal integration

That the synaptic background activity can also modify the temporal integration behavior of the cell is demonstrated in fig. 3. At any particular background frequency $f$, we compute the minimal number of additional excitatory synapses (at $g_{peak} = 0.5\ nS$) necessary to barely generate one action potential. These synapses were chosen randomly from among all excitatory synapses throughout the cell. We compare the case in which all synapses are activated simultaneously (solid line) with the case in which the inputs arrive asynchronously, smeared out over 25 $msec$ (dashed line). If $f = 0$, it requires 115 synapses firing simultaneously to generate a single action potential, while 145 are needed if the input is desynchronized. This small difference between inputs arriving synchronized and at random is due to the long integration period of the cell.

If the background activity increases to $f = 1\ Hz$, 113 synchronized synaptic inputs—spread out all over the cell—are sufficient to fire the cell. If, however, the synaptic input is spread out over 25 $msec$, 202 synapses are now needed in order to trigger a response from the cell. This is mainly due to the much smaller value of $\tau_m$ relative to the period over which the synaptic input is spread out. Note

that the difference in number of simultaneous synaptic inputs needed to fire the cell for $f = 0$ compared to $f = 1$ is small (i.e. 113 vs. 115), in spite of the more than five-fold decrease in somatic input resistance. The effect of the smaller size of the individual EPSP at higher values of $f$ is compensated for by the fact that the resting potential of the cell has been shifted towards the firing threshold of the cell (about $-49$ $mV$).

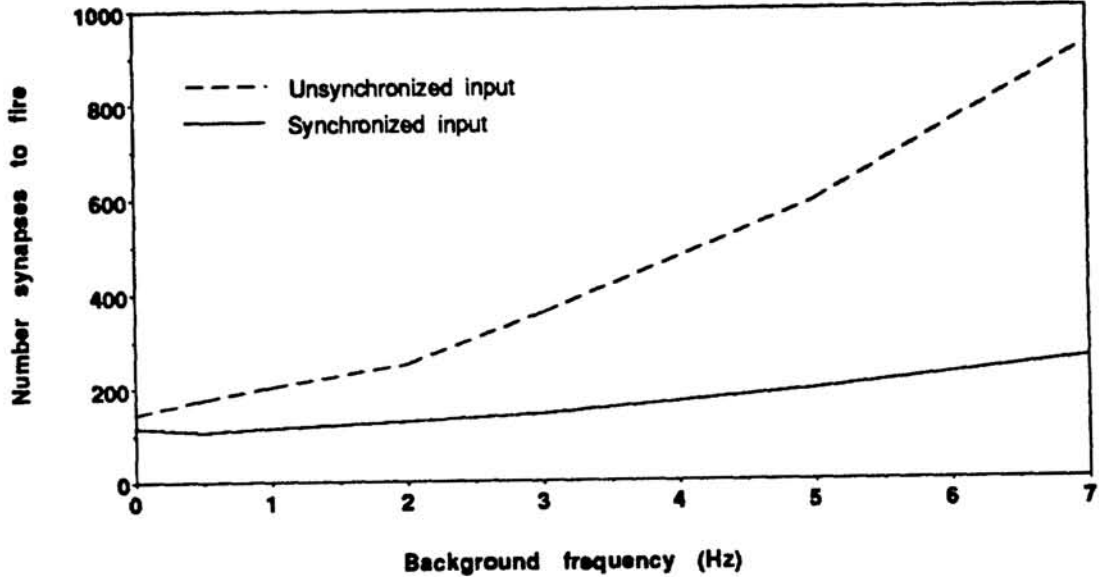

Figure 3: **Phase detection.**
A variable number of excitatory synapses were fired superimposed onto a constant background frequency of 1 $Hz$. They fired either simultaneously (solid line) or spread out in time uniformly during a 25 msec interval (dashed line). The $y$ axis shows the minimum number of synapses necessary to cause the cell to fire.

## 3.4 NMDA synapses

Fast excitatory synaptic input in cortex is mediated by both AMPA or non-NMDA as well as NMDA receptors (Miller et al., 1989). As opposed to the AMPA synapse, the NMDA conductance change depends not only on time but also on the post-synaptic voltage:

$$G(V,t) \;=\; 1.05 \cdot G_{max} \cdot \frac{e^{-t/\tau_1} - e^{-t/\tau_2}}{1 + \eta \cdot [Mg^{2+}] \cdot e^{-\gamma V}}, \tag{1}$$

where $\tau_1 = 40$ $msec$, $\tau_2 = 0.335$ $msec$, $\eta = 0.33$ $mM^{-1}$, $[Mg^{2+}] = 1$ $mM$, $\gamma = 0.06$ $mV^{-1}$. During spontaneous background activity many inputs impinge on the cell and we can time-average the equation above. We will then be left with a purely voltage-dependent conductance.

We measured the somatic input resistance, $R_{in}$, by injecting a small current pulse in the soma (fig. 4) in the standard model. All synapses fired at a 0.5 $Hz$ background frequency. Next we added 4,000 NMDA synapses in addition to the 4,000 non-

NMDA synapses, also at 0.5 $Hz$, and again injected a current pulse. The voltage response is now *larger* by about 65%, corresponding to a *smaller* input conductance, even though we are adding the positive NMDA conductance. This seeming paradox depends on two effects. First, the input conductance is, by definition, $\frac{dI}{dV} = G(V) + \frac{dG(V)}{dV} \cdot (V - E_{rev})$, where G(V) is the conductance specified in eq. (1). For the NMDA synapse this derivative is negative below about $-35\ mV$. Second, due to the excitation the membrane voltage has drifted towards more depolarized values. This will cause a change in the activation of the other voltage-dependent currents. Even though the summed conductance of these active currents will be larger at the new voltage, the derivative $\frac{dI}{dV}$ will be smaller at that point. In other words, activation of NMDA synapses gives a negative contribution to the *input* conductance, even though more conductances have opened up.

Next we replaced $2,000$ of the $4,000$ non-NMDA synapses in the old model with $2,000$ NMDA synapses and recomputed the input resistance as a function of synaptic background activity. The result is overlaid in figure 1a (dashed line). The curve shifts toward larger values of $R_{in}$ for most values of $f$. This shift varies between 50 % – 200 %. The cell is more excitable than before.

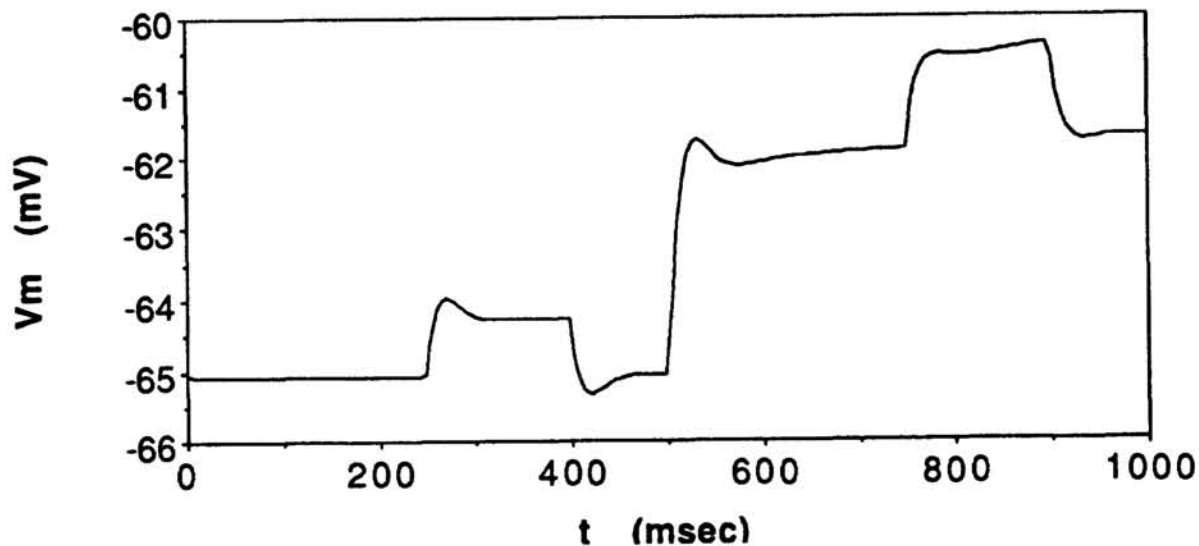

Figure 4: **Negative input conductance from NMDA activation.**
At times $t = 250\ msec$ and $t = 750\ msec$ a 0.05 $nA$ current pulse was injected at the soma and the somatic voltage response was recorded. At $t = 500\ msec$, one NMDA synapse was activated for each non-NMDA synapse, for a total of 8,000 excitatory synaptic inputs. The background frequency was 0.5 $Hz$ for all synapses.

## 4   DISCUSSION

We have seen that parameters such as $R_{in}$, $\tau_m$, and $L$ are not static, but can vary over about one order of magnitude under network control. The potential computational possibilities could be significant.

For example, if a low-contrast stimulus is presented within the receptive field of the cell, the synaptic input rate will be small and the signal-to-noise ratio (SNR) low. In this case, to make the cell more sensitive to the inputs we might want to increase $R_{in}$. This would automatically be achieved as the total network activation is low. We can improve the SNR by integrating over a longer time period, i.e. by increasing $\tau_m$. This would also be a consequence of the reduced network activity. The converse argument can be made for high-contrast stimuli, associated with high overall network activity and low $R_{in}$ and $\tau_m$ values.

Many cortical cells are tuned for various properties of the stimulus, such as orientation, direction, and binocular disparity. As the effective membrane conductance, $G_m$, changes, the tuning curves are expected to change. Depending on the exact circuitry and implementation of the tuning properties, this change in background frequency could take many forms. One example of phase-tuning was given above. In this case the temporal tuning increases with background frequency.

## Acknowledgements

This work was supported by the Office of Naval Research, the National Science Foundation, the James McDonnell Foundation and the International Human Frontier Science Program Organization. Thanks to Tom Tromey for writing the graphic software and to Mike Hines for providing us with NEURON.

## Footnotes

*To whom all correspondence should be addressed.

## References

P. Anderson, M. Raastad & J. F. Storm. (1990) Excitatory synaptic integration in hippocampal pyramids and dentate granule cells. *Symp. Quant. Biol.* **55**, Cold Spring Harbor Press, pp. 81-86.

Ö. Bernander, R. J. Douglas, K. A. C. Martin & C. Koch. (1991) Synaptic background activity influences spatiotemporal integration in single pyramidal cells. *P.N.A.S, USA* **88**: 11569-11573.

R. J. Douglas, K. A. C. Martin & D. Whitteridge. (1991) An intracellular analysis of the visual responses of neurones in cat visual cortex. *J. Physiol.* **440**: 659-696.

M. Hines. (1989) A program for simulation of nerve equations with branching geometries. *Int. J. Biomed. Comput.* **24**: 55-68.

W. R. Holmes & C. D. Woody. (1989) Effects of uniform and non-uniform synaptic activation-distributions on the cable properties of modeled cortical pyramidal neurons. *Brain Research* **505**: 12-22.

A. U. Larkman. (1991) Dendritic morphology of pyramidal neurones of the visual cortex of the rat: III. Spine distributions. *J. Comp. Neurol.* **306**: 332-343.

K. D. Miller, B. Chapman & M. P. Stryker. (1989) Responses of cells in cat visual cortex depend on NMDA receptors. *P.N.A.S.* **86**: 5183-5187.

N. Spruston & D. Johnston. (1992) Perforated patch-clamp analysis of the passive membrane properties of three classes of hippocampal neurons. *J. Neurophysiol.*, in press.